# A rational model of causal induction
# with continuous causes

**Michael D. Pacer**
Department of Psychology
University of California, Berkeley
Berkeley, CA 94720
mpacer@berkeley.edu

**Thomas L. Griffiths**
Department of Psychology
University of California, Berkeley
Berkeley, CA 94720
Tom_Griffiths@berkeley.edu

## Abstract

Rational models of causal induction have been successful in accounting for people's judgments about causal relationships. However, these models have focused on explaining inferences from discrete data of the kind that can be summarized in a $2 \times 2$ contingency table. This severely limits the scope of these models, since the world often provides non-binary data. We develop a new rational model of causal induction using continuous dimensions, which aims to diminish the gap between empirical and theoretical approaches and real-world causal induction. This model successfully predicts human judgments from previous studies better than models of discrete causal inference, and outperforms several other plausible models of causal induction with continuous causes in accounting for people's inferences in a new experiment.

## 1 Introduction

The problem of causal induction is central to science, and is something at which people are remarkably skilled, especially given its apparent difficulty. Understanding how people identify causal relationships has consequently become a challenge taken up by many research programs in cognitive science. One of the most successful of these programs has used rational solutions to the abstract problem of causal induction (in the spirit of [1, 2]) as a source of explanations for people's inferences [3, 4, 5, 6]. However nearly all this research has assumed people have access to categorical information about whether or not a cause or effect is present on a given trial – the sort of information that appears in a $2 \times 2$ contingency table (see Figure 1(a)). Such an assumption may not be valid for many of the causal relationships that we see in the world.

For a simple example of a situation in which a continuous cause is relevant, consider the case of drinking coffee and wakefulness. Clearly, someone who drinks a beverage made by placing a single drop of coffee in a gallon of water will experience no effects of wakefulness, as an insufficient amount of the cause was present. Meanwhile, the diligent graduate student who imbibes upwards of 10 pots of coffee a day will experience a great deal of wakefulness. How much coffee one drinks is closely linked to whether wakefulness occurs – merely knowing that some amount of coffee was drunk is insufficient. And this problem is not relegated to those who wish to titrate their caffeination; many causes exist along continuous dimensions, even if their effects do not (e.g., medicine dosage and recovery, smoking and related death from cancer).[1]

The primary strategy that has been explored in previous work on causal induction from continuous causes is one in which ambiguous examples are immediately categorized as indicating either the presence or the absence of the cause. This approach, taken by Marsh and Ahn [9], provides a way to

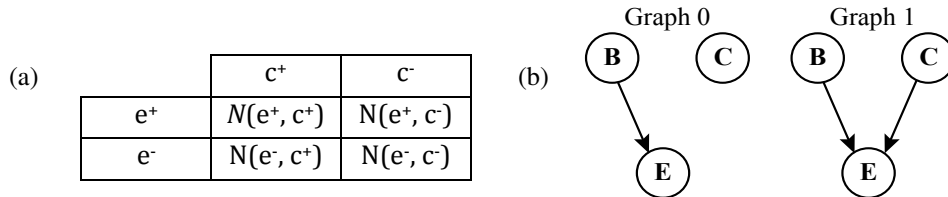

Figure 1: Causal induction. (a) A $2 \times 2$ contingency table. $C$ is the cause, $E$ the effect, with $c^+$ and $c^-$ indicating the presence and absence of the cause, similarly for $e^+$ and $e^-$. (b) Graphical models showing possible causal relationships between cause $C$, effect $E$, and background $B$.

reduce continuous causes to the familiar binary case. In this paper, however, we argue that another approach can be fruitful – developing models that work directly with continuous values. We extend the causal support model [4], originally defined for binary causes, to work with continuous-valued causes. We then re-analyze the results of Marsh and Ahn [9], comparing people's causal judgments to predictions made by a number of rational models of causal induction with both discrete and continuous causes. The predictions made by the continuous models for these experiments perform well, but are extremely similar, which led us to conduct a new experiment using stimuli that discriminate among the different models. We show that continuous causal support provides a better account of these data than the other models we consider.

## 2   Background

In this section we review previous work on rational models of causal induction, and summarize the results of Marsh and Ahn [9] that we will use to evaluate different models later in the paper.

### 2.1   Rational models of causal induction

Rational models of causal induction have focused on the problem of determining the nature of the relationship between a cause $C$ and an effect $E$. These models can be divided into two groups. One group focuses on estimating causal *strength*, such as $\Delta P$ [10], causal power [3] and pCI [11], which attempt to identify the degree of relationship between two variables. The other group focuses on causal *structure*, such as causal support [4], which attempts to identify how certain one can be that a causal relationship exists at all. The causal support model has proven effective in predicting human judgments in several studies [4, 5, 6], and we use it as the starting point for our model of causal induction with continuous causes. The causal support model can be most easily described in the context of causal graphical models [12] (see Figure 1(b)). In particular, we consider two graphical models, Graph 0 ($G_0$) and Graph 1 ($G_1$), and we want to determine the log posterior odds of the models given some data $D$ (i.e. $\log \frac{P(G_1|D)}{P(G_0|D)}$). If we assume that both graphs are equally likely a priori (i.e. $P(G_0) = P(G_1)$), then this is equivalent to calculating the log Bayes factor ($\log \frac{P(D|G_1)}{P(D|G_0)}$). In its most general form causal support is this calculation, described less technically as identifying the evidence that $D$ provides in favor of $G_1$ over $G_0$ [4].

In the particular case of causal inference over binary variables, we have three random variables representing the unknown background causes assumed to be always present ($B$), the possible cause ($C$) and the effect ($E$) in question. In Graph 0 ($G_0$) only $B$ causes $E$, and how often it does so is described by the weight parameter, $w_0$. Thus the probability of the event occurring under $G_0$ is $P(e^+|b^+, w_0; G_0) = w_0$.[2] Graph 1 ($G_1$) allows $C$ to potentially influence the probability of $E$. In particular we say $C$ also has an associated weight parameter $w_1$. How we parameterize the relationship between $B$, $C$, and $E$ determines the type of causal relationship we are considering. In order to capture generative causal relationships we use a noisy-OR parameterization for $P(e|b^+, c, w_0, w_1; G_1)$. That is, under $G_1$ the probability of $E$ occurring (assuming $b^+$) is

$$P(e^+|b^+, c, w_0, w_1; G_1) = 1 - (1 - w_0)(1 - w_1)^c \tag{1}$$

a similar noisy-AND-NOT parameterization can be used for preventive causes [4], but we focus on generative causes in this paper.

Having defined these graphical models, we can compute the corresponding likelihoods. The data consists of the values of all $n$ observed occurrences of cause and effects (i.e. $D = \{(e_1, c_1), (e_2, c_2), \ldots, (e_n, c_n)\}$). Assuming trials are conditionally independent, we have

$$P(D|G_k) = \prod_{i=1}^{n} P(e_i|c_i, b^+, w_0, w_1; G_k) \tag{2}$$

where the noisy-OR parameterization is used, as in Equation 1. If we were concerned with estimating causal strength, we could use this likelihood to determine the estimates of $w_0$ and $w_1$ under $G_1$ and $G_0$. However, if we want to compute a measure of causal *structure* we need to integrate over all possible values of $w_0$ and $w_1$, assuming prior distributions on $w_0$ and $w_1$. In the original causal support model [4], a uniform prior was used on $w_0$ and $w_1$ (for a more complex prior, see [6]).

Despite its success in modeling human judgments, this measure of causal support only works in a limited set of cases – those cases where data can be summarized in a $2\times 2$ contingency table. In order to address more complicated data sets (e.g. continuous-valued causes), significant modifications are needed. These modifications can be made to the model or the data. We propose a modification to the model, while others (e.g., [9]) have attempted to solve this by collapsing continuous data into binary form. We discuss the consequences of the latter strategy in the next section.

## 2.2 Previous work on continuous-valued causal induction

Marsh and Ahn [9] note the insufficiencies of current models of causal induction that result from considering only binary variables. Assuming that the data must be coerced into binary form, they proposed two potential solutions to this problem, and ruled out one of these options. The first solution is that people simply ignore ambiguous information, and only deal with instances that can easily be categorized into "cause" and "not cause". They reject this solution and instead opt for the idea that learners "spontaneously categorize ambiguous evidence into one of the four types of evidence [used in contingency tables]." [9] (p. 4)

To test these claims, Marsh and Ahn conducted a series of experiments in which participants observe visual stimuli (e.g., Figure 2 (a)) representing a particular value along a continuous dimension paired with a (binary) event either occurring or not occurring. Participants were asked to use these images to do two things. First, they were asked to estimate how many examples of each type of data they had seen. Then, participants were asked "to judge the strength between $C$ and $E$ on a scale from 0 (not a cause) to 100 (strongly causes)". Marsh and Ahn used this second measure to show that participants use ambiguous evidence when making causal judgments, refuting the idea that people ignore the instances which cannot be easily categorized. Furthermore, they discovered that engaging in causal inference changes participants' judgments of how many instances of each category they saw. For example, when the "ambiguous" stimuli were paired with the effect (e.g., condition AE of Experiment 1, see Table 1), they found that participants claimed to have seen more examples of the $C$ category. This evidence that people's frequency ratings were altered based on whether or not the effect was paired with the ambiguous stimuli was used to dismiss the possibility that participants were learning a continuous causal relationship.

While Marsh and Ahn demonstrate that causal induction altered how people assigned ambiguous stimuli to categories, this does not necessarily mean that people were spontaneously categorizing these stimuli and using that categorization information to make causal judgments. An alternative account is that the boundary between the categories was ambiguous, and the evidence about the relationship between cause and effect influenced where people placed this boundary. Previous research suggests that category structures should not always be thought of as fixed [13] and that causal information can be used when learning category structures and meanings [14]. Our focus here is on investigating how people might induce causal relationships that involve continuous variables, rather than understanding their influence on categorization. However, the existence of a plausible alternative account of Marsh and Ahn's results raises the possibility that we can understand their data without assuming that people spontaneously categorize ambiguous stimuli in order to make causal judgments. We will explore this possibility after introducing our rational model of causal induction.

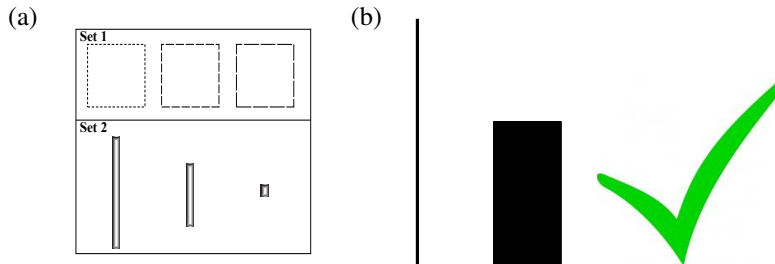

(a)            (b)

Figure 2: Examples of continuous-valued stimuli. (a) Two sets of stimuli used by Marsh and Ahn [9]. The extreme stimuli indicated the presence and absence of a cause, while the intermediate stimulus was deemed "ambiguous". (b) A stimulus used in our experiments.

## 3   Defining causal support for continuous causes

Our goal in this section is to extend the rational analysis used to define the causal support model [4] to causes with continuous values. Following the original model, we take causal support to be the log likelihood ratio in favor of $G_1$ over $G_0$, and assume that the causes combine in a noisy-OR. However, rather than assuming that the influence of $C$ is described by a single parameter $w_1$, we instead define a function($f$) that maps $c$ the value of $C \in \mathbb{R}$ into $[0, 1]$. For any such function $f_\lambda(\cdot) : \mathbb{R} \to [0, 1]$, with parameters $\lambda$, we then have the parameterization

$$P(e^+|b^+, c, w_0, \lambda; G_1) = 1 - (1 - w_0)(1 - f_\lambda(c)) \tag{3}$$

where $c$ is the (continuous) value of the cause $C$. The function $f_\lambda(\cdot)$, thus plays a very similar role to that of the link function in generalized linear models.

We use a specific choice for $f_\lambda(\cdot)$: the probit function (the cumulative distribution function (CDF) of the standard Normal distribution [15]), denoted $\Phi(\cdot)$. The influence of $C$ is encoded in two parameters, a bias parameter $\theta$ and a gain parameter $\gamma$. This gives the full parameterization

$$P(e^+|b^+, c, w_0, \theta, \gamma; G_1) = 1 - (1 - w_0)(1 - \Phi\big(\frac{c - \theta}{\gamma}\big))$$

where $\theta$ indicates the point where the effective strength of $C$ will be 0.5, and $\gamma$ determines the sharpness of the transition in strength around this threshold. It is straightforward to show that the original causal support model corresponds to a special case of this model when $C$ only takes on a single value when it is present.[3] Under the assumption that there is no background rate of occurrence (i.e., $w_0 = 0$), this model is nearly equivalent to probit regression, which provides an excellent comparison case for identifying the role that the noisy-OR plays in explaining people's judgments.

To complete the specification of the model, we need to define prior distributions on the parameters. For the results we report here $w_0 \sim U(0, 1)$, as in [4], and we use the observed values $c^{(n)}$ to produce the priors over $\theta$ and $\gamma$. We take $\theta \sim U(c_{\min}, c_{\max})$, where $c_{\min}$ is the minimum of $c^{(n)}$, and $c_{\max}$ is the maximum. This allows the prior on $\theta$ to be as uninformative as possible while only sampling from the range of values over which inference could be reasonably made. The prior on $\gamma$ is a mixture distribution, where we draw a variable $z$ from an inverse Wishart distribution with one degree of freedom and a mean corresponding to the sample variance, and then set $\gamma$ to either $\sqrt{z}$ or $-\sqrt{z}$ with equal probability. Initial investigations suggest the model is relatively robust to prior choice (e.g. varying the degrees of freedom in the Inverse Wishart does not substantially change model predictions). Because of the complexity of analytically determining the joint likelihood, we use Monte Carlo simulation to approximate the integral over these parameters.

Table 1: Contingencies and mean causal ratings from Marsh and Ahn [9]

| Contingencies | Conditions | | | | | |
|---|---|---|---|---|---|---|
| | Ex1:$AE$ | Ex1:$A\bar{E}$ | Ex2:Zero | Ex2:Weak | Ex2:Moderate | Ex2:Perfect |
| $N(e^+, c^+)$ | 38 | 18 | 10\|10\|10 | 33\|26\|26 | 36\|32\|32 | 40\|40\|40 |
| $N(e^-, c^-)$ | 18 | 38 | 10\|10\|10 | 13\|13\|13 | 16\|16\|16 | 20\|20\|20 |
| $N(e^-, c^+)$ | 2 | 2 | 10\|10\|10 | 7 \| 7 \| 7 | 4 \| 4 \| 4 | 0 \| 0 \| 0 |
| $N(e^+, c^-)$ | 2 | 2 | 10\|10\|10 | 7 \|14\| 7 | 4 \| 8 \| 4 | 0 \| 0 \| 0 |
| Causal Ratings: | 79.2 | 78.5 | 28.3 | 36.2 | 60.6 | 81.0 |

Note: Ex1 and Ex2 refer to Experiments 1 and 2. Vertical bars in Ex2 contingencies separate the three possible strategies (1|2|3) proposed in [9] for assimilating ambiguous stimuli.

We developed this rational model in order to be able to investigate how people engage in causal inference in the case of continuous causes. We proceeded with this investigation in two ways. First, in order to demonstrate the usefulness of considering *any* model of continuous causal inference, we reanalyzed the causal ratings provided by participants in Marsh and Ahn's [9] Experiments 1 and 2. Second, in order to better identify which model best predicts human judgments among the continuous causal models, we conducted a new experiment designed to distinguish between the various rational models.

## 4 Reanalyzing the results of Marsh and Ahn

We applied the continuous causal support model, together with several models of causal induction from discrete data and alternative statistical models for causal induction from continuous data, to two data sets from Marsh and Ahn [9]: the two conditions of Experiment 1 that contained ambiguous stimuli ($AE$ and $A\bar{E}$), and the four conditions of Experiment 2. Contingencies and mean ratings for these experiments are shown in Table 1.

### 4.1 Models

**Discrete models.** Following [4], we evaluated five models of causal induction from discrete data: $\Delta$P [10], causal power [3], pCI [11], (discrete) causal support [4], and the $\chi^2$ statistic. These models were applied to contingencies derived by discretizing the continuous stimuli in three different ways, following the strategies suggested by Marsh and Ahn: (1) if people believe in a generative causal relationship, all ambiguous information should be incorporated into the cause count (i.e. $e^+, c^+$), (2) that people will classify information as being an example of $e^+, c^+$ and $e^+, c^-$ in a way that is proportional to the relationship they infer from the non-"ambiguous" examples, and (3) that people increase $e^+, c^+$ by the same number of "ambiguous" cases as they would under (2), but they do not similarly do this for $e^+, c^-$. Because there are three potential sets of true event counts under the assimilation hypothesis for Experiment 2, in order to analyze the assimilation hypothesis under the best possible case, we will run the discrete models under all three possible methods of assimilation. These three possible ways of assimilating the ambiguous cases are represented in Table 1, as contingencies separated by vertical bars ("|").

**Continuous models.** We also evaluated several models that consider the causal variable to be continuously valued. This includes the causal support model described in the previous section, as well as several traditional models for statistical inference in cases where there is a relationship between continuous and binary variables. Because they are usually used for hypothesis tests about whether or not there is a relationship between a continuous and a binary variable, the two tests we use are probit regression and a two-sample Student $t$-test. The former tests whether there is a relationship between a continuous valued variable mapped to a binary variable, while the latter tests whether there is a relationship between a binary variable mapped to a continuous variable.

Both continuous causal support and the discrete models have the property that with more evidence there is for a cause the larger the positive score produced by the model. We want a similar property to hold for the statistics we obtain from the alternative continuous models. If we treat the two-

Table 2: Correlations of Models Predictions to Human Data and $\alpha$ values

| | Discrete Model Predictions | | | | | | Continuous Model Predictions | | |
| | Possibility 1 | | Possibility 2 | | Possibility 3 | | | | |
| Model: | $r$ | $\alpha$ | $r$ | $\alpha$ | $r$ | $\alpha$ | Model: | $r$ | $\alpha$ |
|---|---|---|---|---|---|---|---|---|---|
| $\Delta$P: | -0.250 | $2\times10^{-4}$ | -0.250 | $2\times10^{-4}$ | -0.250 | $2\times10^{-4}$ | C-Support: | 0.966 | 0.475 |
| Power: | -0.250 | $2\times10^{-4}$ | -0.250 | $2\times10^{-4}$ | -0.250 | $2\times10^{-4}$ | Probit, $|t|$ : | 0.984 | $2\times10^{-4}$ |
| pCI: | -0.035 | 1.100 | -0.035 | 1.100 | 0.239 | 16.142 | Probit, $|\beta|$ : | 0.876 | 0.320 |
| Support : | 0.679 | 154.950 | 0.240 | $2\times10^{-4}$ | 0.679 | 77.350 | $t$-test, $|t|$: | 0.976 | 1.132 |
| $\chi^2$: | 0.679 | $1\times10^{-5}$ | 0.679 | $1\times10^{-5}$ | 0.679 | $1\times10^{-5}$ | $t$-test, $|\beta|$ : | 0.976 | 1.132 |

sample $t$-test as a case of linear regression (with an indicator variable for whether or not the effect occurred as the regressor), we obtain $\beta$ values for both the probit model and the $t$-test model. We can treat these $\beta$ values as estimates of the strength of the relationship between the two variables. Both methods also produce a $t$ statistic, indicating the evidence that $\beta$ is different from zero. We can treat these $t$ values as alternative measures of causal structure. However, the sign of the $\beta$ and $t$ statistics is highly dependent on the particular way the data is represented, so we will use $|\beta|$ and $|t|$ instead.

In their studies, Marsh and Ahn used four types of continuously varying stimuli that differed slightly in the parameters used to create them. We have designed our models such that they are invariant across specification of the dimension, as long as the specification accurately reflects the variance as observed by participants. The parameters used to generate their stimuli, along with the frequencies which each of these values occurred and the associated effects, can be directly plugged into the models to produce predictions. We ran the model over each set of stimulus values, and averaged these four predictions to obtain the final general predictions the means of which were compared to the mean human judgments.

### 4.2 Results

Following [4], model predictions underwent a nonlinear transformation to accommodate nonlinearities in the response scale. This was the transformation $y = \text{sign}(x) * \text{abs}(x)^\alpha$, where $\alpha$ was chosen to maximize the correlation ($r$) between the mean human ratings and mean model predictions across the conditions. The results are shown in Table 2.

The re-analysis supports the idea that people were using continuous values in their causal judgments. The best possible correlation achieved by any discrete model was discrete causal support and $\chi^2$, $r = .679$; this is substantially worse than any of the continuous model correlations. On the other hand, the models of continuous causal inference successfully captured much of the variation in responses, with all the continuous models performing well (all $r > .85$). The Probit $|t|$ model had the best performance, $r = .984$, with Continuous causal support and the $t$-test models not far behind, with $r = .966$ and $r = .976$, respectively.

## 5 Distinguishing between the continuous models

In the previous section, all of the models for continuous causal induction performed well. However, the continuous models all made very similar predictions to one another. As a result, it is difficult to distinguish which model of continuous causal induction people might be using. In order to better determine which of these models most accurately captures human causal induction over continuous dimensions, we need to construct data sets that will result in divergent predictions across the various models.

Because of the noisy-OR parameterization of the generative model, (discrete) causal support predictions are sensitive to the base rate of occurrence while standard statistical tests (e.g., $\chi^2$) lack this sensitivity despite being otherwise good approximations for the rational model [4]. The continuous causal support model also uses a noisy-OR parameterization, meaning that it will also be sensitive

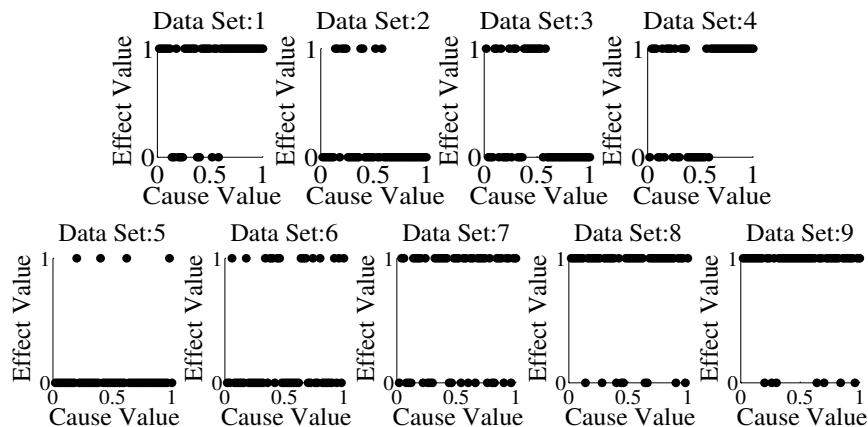

Figure 3: Datasets 1 - 9 for the current experiment. The horizontal axis denotes the value of the cause, while the vertical axis denotes whether or not the event occurred.

to base rates in ways that standard statistical models will not. More generally, the assumption of a particular form for generative causal relationships means that, for some data sets, flipping the values of the effect (replacing a 0 with a 1 and vice versa) can result in different continuous causal support values, though it leaves unchanged the predictions made by the standard methods.

We designed nine data sets to produce such differential predictions. Each data set consisted of a series of fifty $(e, c)$ pairs, where $c \in \{.02, .04, \dots, 1\}$ and $e \in 0, 1$. The only differences between the data sets were the functions defining the relationship between $c$ and $e$. The first four data sets (Figure 3, 1-4) were designed as follows: (1) for $c < .6$ then $e \sim \text{Bern}(.6)$ and for $c \geq .6$ then $e = 1$, (2) flipping the $e$ from (1), (3) for $c < .6$ then $e \sim \text{Bern}(.6)$ and for $c \geq .6$ then $e = 0$, and (4) flipping the $e$ from (3). The next five data sets (Figure 3, 5-9) were meant to be analogous to base rate effects studied in [4]. There was no relationship between the value of $c$ and $e$, but the rate at which $e = 1$ differed between data sets, sampled from $\text{Bern}(p)$ with $p = .1, .25, .5, .75, .9$, for data sets 5-9, respectively. These datasets were then used as the basis for a new behavioral experiment.

## 5.1  Method

**Participants.** A total of 147 participants were recruited through the Amazon Mechanical Turk web service and were paid $0.25 for their participation. Participants were only asked for one such judgment, and were randomly sorted into one of the nine data set conditions we described above. In order to account for any participants who did not read the instructions and consider the data, we eliminated any participants who took less than sixty seconds to complete the study.[4] Because of this constraint, twelve participants were removed, leaving 135 participants for analysis. After removing these participants, we were left with fifteen participants in each condition.

**Procedure.** Participants were told that they would be assisting a scientist in identifying "whether or not different levels of a chemical cause a type of bacteria to produce a protein". They were told that they would see an array of fifty images like the one in Figure 2(b), each of which denoted the outcome of one batch of bacteria. Each of the images consisted of three elements: (1) a black bar that denoted both how much of a chemical was in that batch of bacteria by how large it was with relation to (2) a constant gray line, where a larger bar relative to this indicated that more of the chemical was present, and (3) either a green checkmark or a red cross which denoted whether or not the protein was found. Which images were included in the array were determined by the data condition, and the images were sorted into a random order for each participant before being placed in the array. Participants were told to take their time in analyzing the data, and then were asked to rate "whether they think the chemical causes the protein to be produced" on a 0-100 scale, where 0

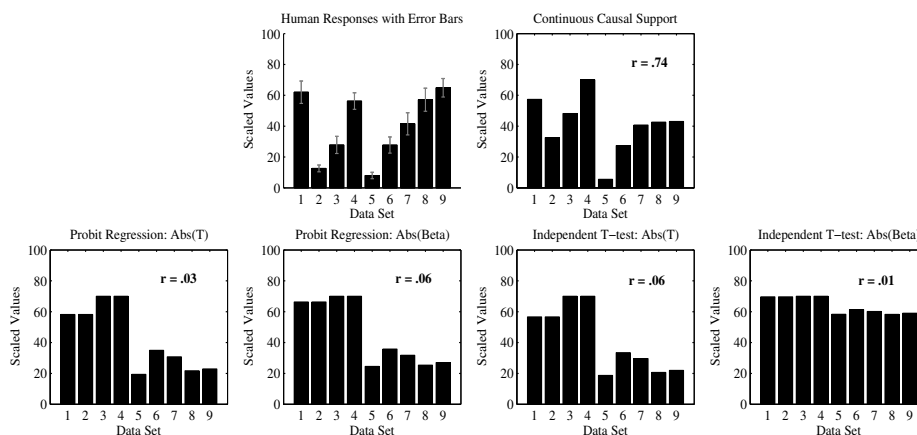

Figure 4: Experimental results, showing human judgments (error bars are one standard error), together with unscaled model predictions and corresponding correlations.

meant *extremely unlikely* and 100 meant *extremely likely*. This scale was designed to obtain scalar estimates of degrees of belief in causal structure [6].

## 5.2 Results

As above, we use a power-law transformation to accommodate nonlinearities in response scale. Before discussing the results, we should note that the Figure 4 does not reflect the maximal correlation between the transformed values of the probit and $t$-test models. The optimized correlation between the mean human responses and mean model predictions for the probit $|\beta|$ model and the $t$-test $|t|$ model were $r = .060$, (with, respectively, $\alpha = 408.6$ and $\alpha = 164.15$ ). The optimized correlation for the probit $|t|$ model was $r = 0.028$ with $\alpha = 2 \times 10^{-4}$. The optimized correlation for the $t$-test $|t|$ was $r = 0.012$, with $\alpha = 12.2$. We did not include the optimized graphs because the optimized mean values for all models save the continuous causal support essentially became binary predictions, and as such they did not convey information about *how* the probit and $t$-test model predictions differed from those made by continuous causal support. The values in Figure 4 reflect the the case where no scaling occurred (i.e. where $\alpha = 1$).

The results are striking in that, though all the models performed well at predicting people's judgments in the Marsh and Ahn studies, all but the continuous causal support model perform poorly here. Continuous causal support outperforms every other model of continuous causal inference ($r = .744$, with $\alpha = 0.92$). Still, it does seem to underestimate human causal ratings in data sets 8 and 9 (see Figure 4), which suggests further investigation of this phenomenon is needed.

## 6 Conclusion

We have proposed a new rational model of causal induction using continuous dimensions, continuous causal support, which aims to be a first step towards filling the gap between existing models of causal induction and real-world cases of causal learning. This model successfully predicts human judgments found in previous work, and outperforms several other plausible models of causal induction with continuous causes. Future work will hopefully continue to bring our models of causal induction ever closer to addressing the problem of real-world causal induction, for example by looking at how temporal information is used in conjunction with traditional statistical information. Consistent with a continuous view of causal induction, we suspect that more work in each of these directions is likely to produce positive results.

**Acknowledgements**: This work was supported by a Berkeley Graduate Fellowship given to MP and grants IIS-0845410 from the National Science Foundation and FA-9550-10-1-0232 from the Air Force Office of Scientific Research to TLG.

## Footnotes

[1]We will focus on the case of continuous causes with binary outcomes. Learning the mapping between continuous variables is known as *function learning* (e.g., [7, 8]).

[2]Following [4], a superscript $+$ indicates the presence of a variable, and a $-$ indicates its absence. We also use $c^+$ and $c^-$ to indicate that $C$ takes the values 1 and 0 respectively.

[3]In our continuous model, we assume the cause is always present but with varying strength. If we allow for the possibility that the cause is absent, and that it has no influence on the effect in such a situation, then we obtain $P(e^+|b^+, c^-, w_0, \theta, \gamma; G_1) = w_0$, as required. We then observe that $\Phi\big(\frac{c-\theta}{\gamma}\big)$ plays an analogous role in Equation 3 to $w_1$ in (1). To show equivalence, we need to show that it is possible for this quantity to have a uniform prior when $c = 1$. Take $\gamma = 1$, and define a Gaussian prior on $\theta$ with mean 1 and unit variance. $\frac{c-\theta}{\gamma}$ then follows a Gaussian distribution with mean 0 and unit variance. Since $\Phi(\cdot)$ is the CDF of the standard Normal, the distribution of $\Phi\big(\frac{c-\theta}{\gamma}\big)$ is uniform on $[0, 1]$.

[4]Though we eliminated these subjects from the analysis here, not eliminating them does not change any of the $r$ scores by more than $\pm.02$. In fact, including these participants increases the performance of our model and decreases the performance of the alternative models.

# References

[1] J. R. Anderson. *The adaptive character of thought*. Erlbaum, Hillsdale, NJ, 1990.

[2] D. Marr. *Vision*. W. H. Freeman, San Francisco, CA, 1982.

[3] P. Cheng. From covariation to causation: A causal power theory. *Psychological Review*, 104:367–405, 1997.

[4] T. L. Griffiths and J. B. Tenenbaum. Structure and strength in causal induction. *Cognitive Psychology*, 51:354–384, 2005.

[5] T. L. Griffiths and J. B. Tenenbaum. Theory-based causal induction. *Psychological review*, 116(4):661, 2009.

[6] H. Lu, A. L. Yuille, M. Liljeholm, P. W. Cheng, and K. J. Holyoak. Bayesian generic priors for causal learning. *Psychological review*, 115(4):955, 2008.

[7] J. R. Busemeyer, E. Byun, E. L. DeLosh, and M. A. McDaniel. Learning functional relations based on experience with input-output pairs by humans and artificial neural networks. In K. Lamberts and D. Shanks, editors, *Concepts and Categories*, pages 405–437. MIT Press, Cambridge, 1997.

[8] T. L. Griffiths, C. G. Lucas, J. J. Williams, and M. L. Kalish. Modeling human function learning with gaussian processes. In Daphne Koller, Yoshua Bengio, Dale Schuurmans, and Léon Bottou, editors, *Advances in Neural Information Processing Systems*, volume 21, Cambridge, MA, 2009. MIT Press.

[9] J. K. Marsh and W. Ahn. Spontaneous assimilation of continuous values and temporal information in causal induction. *Journal of Experimental Psychology: Learning, Memory, and Cognition*, 35(2):334, 2009.

[10] P. W. Cheng and L. R. Novick. A probabilistic contrast model of causal induction. *Journal of Personality and Social Psychology*, 58:545–567, 1990.

[11] P. A. White. Making causal judgments from the proportion of confirming instances: the pCI rule. *Journal of Experimental Psychology: Learning, Memory, and Cognition*, 29:710–727, 2003.

[12] J. Pearl. *Probabilistic reasoning in intelligent systems*. Morgan Kaufmann, San Francisco, CA, 1988.

[13] M. R. Waldmann and Y. Hagmayer. Categories and causality: The neglected direction. *Cognitive Psychology*, 53(1):27–58, 2006.

[14] M. R. Waldmann, K. J. Holyoak, and A. Fratianne. Causal models and the acquisition of category structure. *Journal of Experimental Psychology: General*, 124:181–206, 1995.

[15] C. I. Bliss. The calculation of the dosage-mortality curve. *Annals of Applied Biology*, 22(1):134–167, 1935.

